# STDP enables spiking neurons to detect hidden causes of their inputs

**Bernhard Nessler, Michael Pfeiffer, and Wolfgang Maass**
Institute for Theoretical Computer Science, Graz University of Technology
A-8010 Graz, Austria
{nessler,pfeiffer,maass}@igi.tugraz.at

## Abstract

The principles by which spiking neurons contribute to the astounding computational power of generic cortical microcircuits, and how spike-timing-dependent plasticity (STDP) of synaptic weights could generate and maintain this computational function, are unknown. We show here that STDP, in conjunction with a stochastic soft winner-take-all (WTA) circuit, induces spiking neurons to generate through their synaptic weights implicit internal models for subclasses (or "causes") of the high-dimensional spike patterns of hundreds of pre-synaptic neurons. Hence these neurons will fire after learning whenever the current input best matches their internal model. The resulting computational function of soft WTA circuits, a common network motif of cortical microcircuits, could therefore be a drastic dimensionality reduction of information streams, together with the autonomous creation of internal models for the probability distributions of their input patterns. We show that the autonomous generation and maintenance of this computational function can be explained on the basis of rigorous mathematical principles. In particular, we show that STDP is able to approximate a stochastic online Expectation-Maximization (EM) algorithm for modeling the input data. A corresponding result is shown for Hebbian learning in artificial neural networks.

## 1 Introduction

It is well-known that synapses change their synaptic efficacy ("weight") $w$ in dependence of the difference $t_{post} - t_{pre}$ of the firing times of the post- and presynaptic neuron according to variations of a generic STDP rule (see [1] for a recent review). However, the computational benefit of this learning rule is largely unknown [2, 3]. It has also been observed that local WTA-circuits form a common network-motif in cortical microcircuits [4]. However, it is not clear how this network-motif contributes to the computational power and adaptive capabilities of laminar cortical microcircuits, out of which the cortex is composed. Finally, it has been conjectured for quite some while, on the basis of theoretical considerations, that the discovery and representation of hidden causes of their high-dimensional afferent spike inputs is a generic computational operation of cortical networks of neurons [5]. One reason for this belief is that the underlying mathematical framework, Expectation-Maximization (EM), arguably provides the most powerful approach to unsupervised learning that we know of. But one has so far not been able to combine these three potential pieces (STDP, WTA-circuits, EM) of the puzzle into a theory that could help us to unravel the organization of computation and learning in cortical networks of neurons.

We show in this extended abstract that STDP in WTA-circuits approximates EM for discovering hidden causes of large numbers of input spike trains. We first demonstrate this in section 2 in an application to a standard benchmark dataset for the discovery of hidden causes. In section 3 we show that the functioning of this demonstration can be explained on the basis of EM for simpler non-spiking approximations to the spiking network considered in section 2.

## 2 Discovery of hidden causes for a benchmark dataset

We applied the network architecture shown in Fig. 1A to handwritten digits from the MNIST dataset [6].[1] This dataset consists of $70,000$ $28 \times 28$-pixel images of handwritten digits[2], from which we picked the subset of $20,868$ images containing only the digits $0, 3$ and $4$. Training examples were randomly sampled from this subset with a uniform distribution of digit classes.

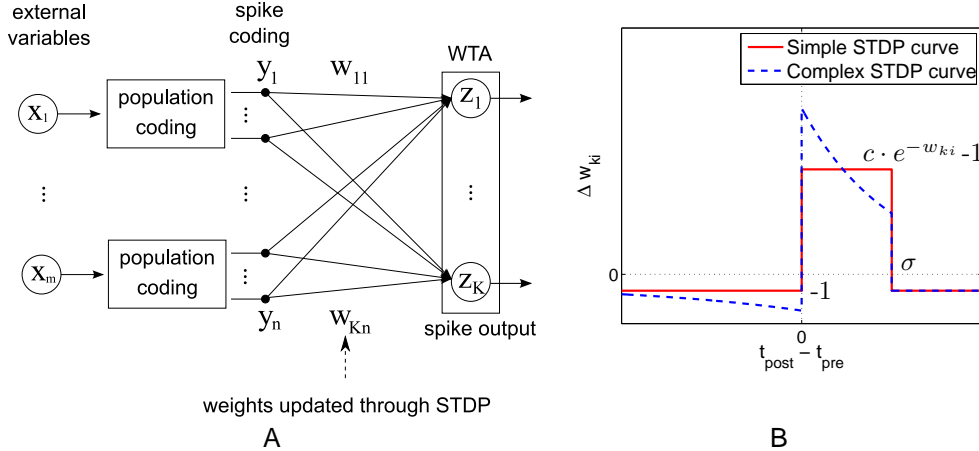

Figure 1: **A)** Architecture for learning with STDP in a WTA-network of spiking neurons. **B)** Learning curve for the two STDP rules that were used (with $\sigma = 10$ms). The synaptic weight $w_{ki}$ is changed in dependence of the firing times $t_{pre}$ of the presynaptic neuron $y_i$ and $t_{post}$ of the post-synaptic neuron $z_k$. If $z_k$ fires at time $t$ without a firing of $y_i$ in the interval $[t - \sigma, t + 2\sigma]$, $w_{ki}$ is reduced by $1$. The resulting weight change is in any case multiplied with the current learning rate $\eta$, which was chosen in the simulations according to the variance tracking rule[7].

Pixel values $x_j$ were encoded through population coding by binary variables $y_i$ (spikes were produced for each variable $y_i$ by a Poisson process with a rate of 40 Hz for $y_i = 1$, and 0 Hz for $y_i = 0$, at a simulation time step of 1ms, see Fig. 2A). Every training example $\mathbf{x}$ was presented for 50ms. Every neuron $y_i$ was connected to all $K = 10$ output neurons $z_1, \ldots, z_{10}$. A Poisson process caused firing of one of the neurons $z_k$ on average every 5ms (see [8] for a more realistic firing mechanism). The WTA-mechanism ensured that only one of the output neurons could fire at any time step. The winning neuron at time step $t$ was chosen from the soft-max distribution

$$p(z_k \text{ fires at time } t | \mathbf{y}) = \frac{e^{u_k(t)}}{\sum_{l=1}^{K} e^{u_l(t)}}, \tag{1}$$

where $u_k(t) = \sum_{i=1}^{n} w_{ki} \tilde{y}_i(t) + w_{k0}$ represents the current membrane potential of neuron $z_k$ (with $\tilde{y}_i(t) = 1$ if $y_i$ fired within the time interval $[t - 10\text{ms}, t]$, else $\tilde{y}_i(t) = 0$).[3]

STDP with the learning curves shown in Fig. 1B was applied to all synapses $w_{ki}$ for an input consisting of a continuous sequence of spike encodings of handwritten digits, each presented for 50ms (see

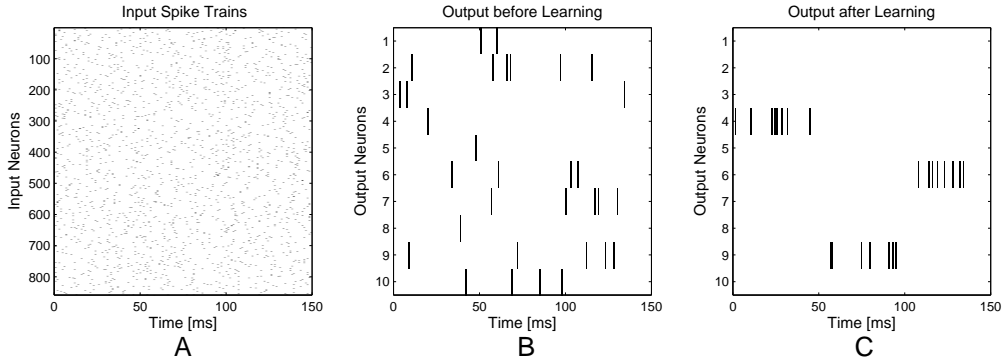

Figure 2: Unsupervised classification learning and sparsification of firing of output neurons after training. For testing we presented three examples from an independent test set of handwritten digits $0, 3, 4$ from the MNIST dataset, and compared the firing of the output-neurons before and after learning. **A)** Representation of the three handwritten digits $0, 3, 4$ for 50ms each by 858 spiking neurons $y_i$. **B)** Response of the output neurons before training. **C)** Response of the output neurons after STDP (according to Fig. 1B) was applied to their weights $w_{ki}$ for a continuous sequence of spike encodings of 4000 randomly drawn examples of handwritten digits $0, 3, 4$, each represented for 50ms (like in panel A). The three output neurons $z_4, z_9, z_6$ that respond have generated internal models for the three shown handwritten digits according to Fig. 3C.

Fig. 2A).[4] The learning rate $\eta$ was chosen locally according to the variance tracking rule[7]. Fig. 2C shows that for subsequent representations of new handwritten samples of the same digits only one neuron responds during each of the 50ms while a handwritten digit is shown. The implicit internal models which the output neurons $z_1, \ldots, z_{10}$ had created in their weights after applying STDP are made explicit in Fig. 3B and C. Since there were more output neurons than digits, several output neurons created internal models for different ways of writing the same digit. When after applying STDP to 2000 random examples of handwritten digits 0 and 3 also examples of handwritten digit 4 were included in the next 2000 examples, the internal models of the 10 output neurons reorganized autonomously, to include now also two internal models for different ways of writing the digit 4. The adaptation of the spiking network to the examples shown so far is measured in Fig. 3A by the normalized conditional entropy $H(L|Z)/H(L, Z)$, where $L$ denotes the correct classification of each handwritten digit $\mathbf{y}$, and $Z$ is the random variable which denotes the cluster assignment with $p(Z = k|\mathbf{y}) = p(z_k = 1|\mathbf{y})$, the firing probabilities at the presentation of digit $\mathbf{y}$, see (1).

Since after training by STDP each of the output neurons fire preferentially for one digit, we can measure the emergent classification capability of the network. The resulting weight-settings achieve a classification error of $2.19\%$ on the digits 0 and 3 after 2000 training steps and $3.68\%$ on all three digits after 4000 training steps on independent test sets of 10,000 new samples each.

## 3 Underlying theoretical principles

We show in this section that one can analyze the learning dynamics of the spiking network considered in the preceding section (with the simple STDP curve of Fig. 1B with the help of Hebbian learning (using rule (12)) in a corresponding non-spiking neural network $\mathcal{N}_{\mathbf{w}}$. $\mathcal{N}_{\mathbf{w}}$ is a stochastic artificial neural network with the architecture shown in Fig. 1A, and with a parameter vector $\mathbf{w}$ consisting of thresholds $w_{k0}$ $(k = 1, \ldots, K)$ for the $K$ output units $z_1, \ldots, z_K$ and weights $w_{ki}$ for the connection from the $i^{\text{th}}$ input node $y_i$ $(i = 1, \ldots, n)$ to the $k^{\text{th}}$ output unit $z_k$. We assume that this network receives at each discrete time step a binary input vector $\mathbf{y} \in \{0, 1\}^n$ and outputs a binary vector $\mathbf{z} \in \{0, 1\}^K$ with $\sum_{k=1}^{K} z_k = 1$, where the $k$ such that $z_k = 1$ is drawn from the distribution

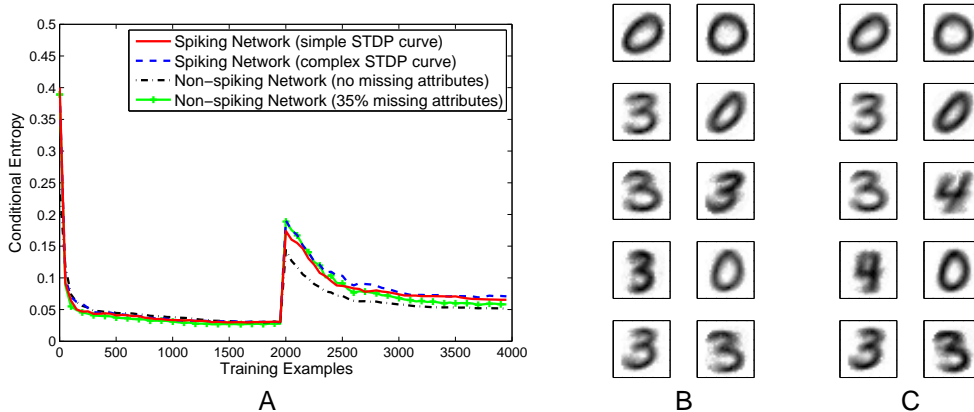

A                                B              C

Figure 3: Analysis of the learning progress of the spiking network for the MNIST dataset. **A)** Normalized conditional entropy (see text) for the spiking network with the two variants of STDP learning rules illustrated in Fig. 1B (red solid and blue dashed lines), as well as two non-spiking approximations of the network with learning rule (12) that are analyzed in section 3. According to this analysis the non-spiking network with $35\%$ missing attributes (dash-dotted line) is expected to have a very similar learning behavior to the spiking network. 2000 random examples of handwritten digits 0 and 3 were presented (for 50ms each) to the spiking network as the first 2000 examples. Then for the next 2000 examples also samples of handwritten digit 4 were included. **B)** The implicit internal models created by the neurons after 2000 training examples are made explicit by drawing for each pixel the difference $w_{ki} - w_{k(i+1)}$ of the weights for input $y_i$ and $y_{i+1}$ that encode the two possible values (black/white) of the variable $x_j$ that encodes this pixel value. One can clearly see that neurons created separate internal models for different ways of writing the two digits 0 and 3. **C)** Re-organized internal models after 2000 further training examples that included digit 4. Two output neurons had created internal models for the newly introduced digit 4.

over $\{1, \ldots, K\}$ defined by

$$p(z_k = 1|\mathbf{y}, \mathbf{w}) = \frac{e^{u_k}}{\sum\limits_{l=1}^{K} e^{u_l}} \qquad \text{with} \quad u_k = \sum_{i=1}^{n} w_{ki}\, y_i + w_{k0} \ . \tag{2}$$

We consider the case where there are arbitrary discrete external variables $x_1, \ldots, x_m$, each ranging over $\{1, \ldots, M\}$ (we had $M = 2$ in section 2), and assume that these are encoded through binary variables $y_1, \ldots, y_n$ for $n = m \cdot M$ with $\sum_{i=1}^{n} y_i = m$ according to the rule

$$y_{(j-1)\cdot M+r} = 1 \quad \Longleftrightarrow \quad x_j = r \ , \qquad \text{for } j = 1, \ldots, m \text{ and } r = 1, \ldots, M. \tag{3}$$

In other words: the group $G_j$ of variables $y_{(j-1)\cdot M+1}, \ldots, y_{(j-1)\cdot M+M}$ provides a population coding for the discrete variable $x_j$.

We now consider a class of probability distributions that is particularly relevant for our analysis: mixtures of multinomial distributions [9], a generalization of mixtures of Bernoulli distributions (see section 9.3.3 of [10]). This is a standard model for latent class analysis [11] in the case of discrete variables. Mixtures of multinomial distributions are arbitrary mixtures of $K$ distributions $p_1(\mathbf{x}), \ldots, p_K(\mathbf{x})$ that factorize, i.e.,

$$p_k(\mathbf{x}) = \prod_{j=1}^{m} p_{kj}(x_j)$$

for arbitrary distributions $p_{kj}(x_j)$ over the range $\{1, \ldots, M\}$ of possible values for $x_j$. In other words: there exists some distribution over hidden binary variables $z_k$ with $\sum_{k=1}^{K} z_k = 1$, where the $k$ with $z_k = 1$ is usually referred to as a hidden "cause" in the generation of $\mathbf{x}$, such that

$$p(\mathbf{x}) = \sum_{k=1}^{K} p(z_k = 1) \cdot p_k(\mathbf{x}). \tag{4}$$

We first observe that any such distribution $p(\mathbf{x})$ can be represented with some suitable weight vector $\mathbf{w}$ by the neural network $\mathcal{N}_{\mathbf{w}}$, after recoding of the multinomial variables $x_j$ by binary variables $y_i$ as defined before:

$$p(\mathbf{y}|\mathbf{w}) = \sum_{k=1}^{K} e^{u_k^*} \qquad \text{with} \qquad u_k^* := \sum_{i=1}^{n} w_{ki}^* \, y_i + w_{k0}^* \quad , \tag{5}$$

for

$$w_{ki}^* := \log p(y_i = 1 | z_k = 1) \qquad \text{and} \qquad w_{k0}^* := \log p(z_k = 1) \quad . \tag{6}$$

In addition, $\mathcal{N}_{\mathbf{w}}$ defines for any weight vector $\mathbf{w}$ whose components are normalized, i.e.

$$\sum_{k=1}^{K} e^{w_{k0}} = 1 \quad \text{and} \quad \sum_{i \in G_j} e^{w_{ki}} = 1 \, , \qquad \text{for } j = 1, \dots, m;\, k = 1, \dots, K, \tag{7}$$

a mixture of multinomials of the type (4).

The problem of learning a generative model for some arbitrarily given input distribution $p^*(\mathbf{x})$ (or $p^*(\mathbf{y})$ after recoding according to (3)), by the neural network $\mathcal{N}_{\mathbf{w}}$ is to find a weight vector $\mathbf{w}$ such that $p(\mathbf{y}|\mathbf{w})$ defined by (5) models $p^*(\mathbf{y})$ as accurately as possible. As usual, we quantify this goal by demanding that

$$\mathrm{E}_{p^*}[\log p(\mathbf{y}|\mathbf{w})] \tag{8}$$

is maximized.

Note that the architecture $\mathcal{N}_{\mathbf{w}}$ is very useful from a functional point of view, because if (7) holds, then the weighted sum $u_k$ at its unit $z_k$ has according to (2) the value $\log p(z_k = 1|\mathbf{y}, \mathbf{w})$, and the stochastic WTA rule of $\mathcal{N}_{\mathbf{w}}$ picks the "winner" $k$ with $z_k = 1$ from this internally generated model $p(z_k = 1|\mathbf{y}, \mathbf{w})$ for the actual distribution $p^*(z_k = 1|\mathbf{y})$ of hidden causes. We will not enforce the normalization (7) explicitly during the subsequently considered learning process, but rather use a learning rule (12) that turns out to automatically approximate such normalization in the limit.

Expectation Maximization (EM) is the standard method for maximizing $\mathrm{E}_{p^*}[\log p(\mathbf{y}|\mathbf{w})]$. We will show that the simple STDP-rule of Fig. 1B for the spiking network of section 2 can be viewed as an approximation to an online version of this EM method. We will first consider in section 3.1 the standard EM-approach, and show that the Hebbian learning rule (12) provides a stochastic approximation to the maximization step.

## 3.1 Reduction to EM

The standard method for maximizing the expected log-likelihood $\mathrm{E}_{p^*}[\log p(\mathbf{y}|\mathbf{w})]$ with a distribution $p$ of the form $p(\mathbf{y}|\mathbf{w}) = \sum_{\mathbf{z}} p(\mathbf{y}, \mathbf{z}|\mathbf{w})$ with hidden variables $\mathbf{z}$, is to observe that $\mathrm{E}_{p^*}[\log p(\mathbf{y}|\mathbf{w})]$ can be written for arbitrary distributions $q(\mathbf{z}|\mathbf{y})$ in the form

$$\mathrm{E}_{p^*}[\log p(\mathbf{y}|\mathbf{w})] = \mathcal{L}(q, \mathbf{w}) + \mathrm{E}_{p^*}[\mathrm{KL}(q(\mathbf{z}|\mathbf{y})||p(\mathbf{z}|\mathbf{y}, \mathbf{w}))] \tag{9}$$

$$\mathcal{L}(q, \mathbf{w}) = \mathrm{E}_{p^*}\left[ \sum_{\mathbf{z}} q(\mathbf{z}|\mathbf{y}) \log \frac{p(\mathbf{y}, \mathbf{z}|\mathbf{w})}{q(\mathbf{z}|\mathbf{y})} \right] \quad , \tag{10}$$

where $\mathrm{KL}(.)$ denotes the Kullback-Leibler divergence.

In the $E$-step one sets $q(\mathbf{z}|\mathbf{y}) = p(\mathbf{z}|\mathbf{y}, \mathbf{w}^{old})$ for the current parameter values $\mathbf{w} = \mathbf{w}^{old}$, thereby achieving $\mathrm{E}_{p^*}[\mathrm{KL}(q(\mathbf{z}|\mathbf{y})||p(\mathbf{z}|\mathbf{y}, \mathbf{w}^{old}))] = 0$. In the $M$-step one replaces $\mathbf{w}^{old}$ by new parameters $\mathbf{w}$ that maximize $\mathcal{L}(q, \mathbf{w})$ for this distribution $q(\mathbf{z}|\mathbf{y})$. One can easily show that this is achieved by setting

$$w_{ki}^* = \log p^*(y_i = 1 | z_k = 1), \qquad \text{and} \qquad w_{k0}^* = \log p^*(z_k = 1), \tag{11}$$

with values for the variables $z_k$ generated by $q(\mathbf{z}|\mathbf{y}) = p(\mathbf{z}|\mathbf{y}, \mathbf{w}^{old})$, while the values for the variables $\mathbf{y}$ are generated by the external distribution $p^*$. Note that this distribution of $\mathbf{z}$ is exactly the distribution (2) of the output of the neural network $\mathcal{N}_{\mathbf{w}}$ for inputs $\mathbf{y}$ generated by $p^*$.[5] In the following section we will show that this $M$-step can be approximated by applying iteratively a simple Hebbian learning rule to the weights $\mathbf{w}$ of the neural network $\mathcal{N}_{\mathbf{w}}$.

## 3.2 A Hebbian learning rule for the M-step

We show here that the target weight values (11) are the only equilibrium points of the following Hebbian learning rule:

$$\Delta w_{ki} = \begin{cases} \eta\left(e^{-w_{ki}} - 1\right), & \text{if } y_i{=}1 \text{ and } z_k{=}1 \\ -\eta, & \text{if } y_i{=}0 \text{ and } z_k{=}1 \\ 0, & \text{if } z_k = 0, \end{cases} \qquad \Delta w_{k0} = \begin{cases} \eta\left(e^{-w_{k0}} - 1\right), & \text{if } z_k{=}1 \\ -\eta, & \text{if } z_k{=}0 \end{cases} \qquad (12)$$

It is obvious (using for the second equivalence the fact that $y_i$ is a binary variable) that

$$
\begin{aligned}
\mathrm{E}[\Delta w_{ki}] = 0 \;\Leftrightarrow\;\; & p^*(y_i{=}1|z_k{=}1)\eta(e^{-w_{ki}} - 1) - p^*(y_i{=}0|z_k{=}1)\eta = 0 \\
\Leftrightarrow\;\; & p^*(y_i{=}1|z_k{=}1)(e^{-w_{ki}} - 1) + p^*(y_i{=}1|z_k{=}1) - 1 = 0 \\
\Leftrightarrow\;\; & p^*(y_i{=}1|z_k{=}1)e^{-w_{ki}} = 1 \\
\Leftrightarrow\;\; & w_{ki} = \log p^*(y_i{=}1|z_k{=}1) \quad .
\end{aligned}
\qquad (13)
$$

Analogously one can show that $\mathrm{E}[\Delta w_{k0}] = 0 \Leftrightarrow w_{k0} = \log p^*(z_k{=}1)$. With similar elementary calculations one can show that $\mathrm{E}[\Delta w_{ki}]$ has for any $\mathbf{w}$ a value that moves $w_{ki}$ in the direction of $w_{ki}^*$ (in fact, exponentially fast).

One can actually show that one single step of (12) is a linear approximation of the ideal incremental update of $w_{ki} = \log \frac{a_{ki}}{N_k}$, with $a_{ki}$ and $N_k$ representing the values of the corresponding sufficient statistics, as $\log \frac{a_{ki}+1}{N_k+1} = w_{ki} + \log(1 + \eta e^{-w_{ki}}) - \log(1 + \eta)$ for $\eta = \frac{1}{N_k}$. This also reveals the role of the learning rate $\eta$ as the reciprocal of the equivalent sample size[6].

In order to guarantee the stochastic convergence (see [12]) of the learning rule one has to use a decaying learning rate $\eta^{(t)}$ such that $\sum_{t=1}^{\infty} \eta^{(t)} = \infty$ and $\sum_{t=1}^{\infty} (\eta^{(t)})^2 = 0$.[7]

The learning rule (12) is similar to a rule that had been introduced in [13] in the context of supervised learning and reinforcement learning. That rule had satisfied an equilibrium condition similar to (13). But to the best of our knowledge, such type of rule has so far not been considered in the context of unsupervised learning.

One can easily see the correspondence between the update of $w_{ki}$ in (12) and in the simple STDP rule of Fig. 1B. In fact, if each time where neuron $z_k$ fires in the spiking network, each presynaptic neuron $y_i$ that currently has a high firing rate has fired within the last $\sigma = 10\text{ms}$ before the firing of $z_k$, the two learning rules become equivalent. However since the latter condition could only be achieved with biologically unrealistic high firing rates, we need to consider in section 3.4 the case for the non-spiking network where some attributes are missing (i.e., $y_i = 0$ for all $i \in G_j$; for some group $G_j$ that encodes an external variable $x_j$ via population coding).

We first show that the Hebbian learning rule (12) is also meaningful in the case of online learning of $\mathcal{N}_\mathbf{w}$, which better matches the online learning process for the spiking network.

## 3.3 Stochastic online EM

The preceding arguments justify an application of learning rule (12) for a number of steps within each M-step of a batch EM approach for maximizing $\mathrm{E}_p^*[\log p(\mathbf{y}|\mathbf{w})]$. We now show that it is also meaningful to apply the same rule (12) in an online stochastic EM approach (similarly as in [14]), where at each combined EM-step only one example $\mathbf{y}$ is generated by $p^*$, and the learning rule (12)

is applied just once (for $z_k$ resulting from $p(\mathbf{z}|\mathbf{y}, \mathbf{w})$ for the current weights $\mathbf{w}$, or simpler: for the $z_k$ that is output by $\mathcal{N}_{\mathbf{w}}$ for the current input $\mathbf{y}$).

Our strategy for showing that a single application of learning rule (12) is expected to provide progress in an online EM-setting is the following. We consider the Lagrangian $F$ for maximizing $\mathrm{E}_{p^*}[\log p(\mathbf{y}|\mathbf{w})]$ under the constraints (7), and show that an application of rule (12) is expected to increase the value of $F$. We set

$$F(\mathbf{w}, \boldsymbol{\lambda}) = \mathrm{E}_{p^*}[\log p(\mathbf{y}|\mathbf{w})] - \lambda_0 \left( 1 - \sum_{k=1}^{K} e^{w_{k0}} \right) - \sum_{k=1}^{K} \sum_{j=1}^{m} \lambda_{kj} \left( 1 - \sum_{i \in G_j} e^{w_{ki}} \right). \qquad (14)$$

According to (5) one can write $p(\mathbf{y}|\mathbf{w}) = \sum_{k=1}^{K} e^{u_k}$ for $u_k = \sum_{i=1}^{K} w_{ki} \, y_i + w_{k0}$. Hence one arrives at the following conditions for the Lagrange multipliers $\boldsymbol{\lambda}$:

$$\sum_{k=1}^{K} \frac{\partial F}{\partial w_{k0}} = \sum_{k=1}^{K} \left( \mathrm{E}_{p^*}[\frac{e^{u_k}}{\sum_{l=1}^{K} e^{u_l}}] - \lambda_0 e^{w_{k0}} \right) = 0 \qquad (15)$$

$$\sum_{i \in G_j} \frac{\partial F}{\partial w_{ki}} = \sum_{i \in G_j} \left( \mathrm{E}_{p^*}[y_i \, \frac{e^{u_k}}{\sum_{l=1}^{K} e^{u_l}}] - \lambda_{kj} e^{w_{ki}} \right) = 0, \qquad (16)$$

which yield $\lambda_0 = 1$ and $\lambda_{kj} = \mathrm{E}_{p^*}[\frac{e^{u_k}}{\sum_{l=1}^{K} e^{u_l}}]$.

Plugging these values for $\boldsymbol{\lambda}$ into $\nabla_{\mathbf{w}} F \cdot \mathrm{E}_p^*[\Delta\mathbf{w}]$ with $\Delta\mathbf{w}$ defined by (12) shows that this vector product is always positive. Hence even a single application of learning rule (12) to a single new example $\mathbf{y}$, drawn according to $p^*$, is expected to increase $\mathrm{E}_{p^*}[\log p(\mathbf{y}|\mathbf{w})]$ under the constraints (7).

### 3.4 Impact of missing attributes

We had shown at the end of 3.2 that learning in the spiking network corresponds to learning in the non-spiking network $\mathcal{N}_{\mathbf{w}}$ with missing attributes. A profound analysis of the correct handling of missing attribute values in EM can be found in [15]. Their analysis implies that the correct learning action is then not to change the weights $w_{ki}$ for $i \in G_j$. However the STDP rule of Fig. 1B, as well as (12), reduce also these weights by $\eta$ if $z_k$ fires. This yields a modification of the equilibrium analysis (13):

$$\mathrm{E}[\Delta w_{ki}] = 0 \iff (1 - r) \left( p^*(y_i{=}1|z_k{=}1)\eta(e^{-w_{ki}} - 1) - p^*(y_i{=}0|z_k{=}1)\eta \right) - r\eta = 0$$
$$\iff \qquad w_{ki} = \log p^*(y_i{=}1|z_k{=}1) + \log(1 - r) \quad , \qquad (17)$$

where $r$ is the probability that $i$ belongs to a group $G_j$ where the value of $x_j$ is missing. Since this probability $r$ is independent of the neuron $z_k$ and also independent of the current value of the external variable $x_i$, this offset of $\log(1 - r)$ is expected to be the same for all weights. It can easily be verified, that such an offset does not change the resulting probabilities of the competition in the E-step according to (2).

### 3.5 Relationship between the spiking and the non-spiking network

As indicated at the end of section 3.2, the learning process for the spiking network from section 2 with the simple STDP curve from Fig. 1B (and external variables $x_j$ encoded by input spike trains from neurons $y_i$) is equivalent to a somewhat modified learning process of the non-spiking network $\mathcal{N}_{\mathbf{w}}$ with the Hebbian learning rule (12) and external variables $x_j$ encoded by binary variables $y_i$. Each firing of a neuron $z_k$ at some time $t$ corresponds to a discrete time step in $\mathcal{N}_{\mathbf{w}}$ with an application of the Hebbian learning rule (12). Each neuron $y_i$ that had fired during the time interval $[t - 10\text{ms}, t]$ contributes a value $\tilde{y}_i(t) = 1$ to the membrane potential $u_k(t)$ of the neuron $z_k$ at time $t$, and a value $\tilde{y}_i(0) = 0$ if it did not fire during $[t - 10\text{ms}, t]$. Hence the weight updates at time $t$ according to the simple STDP curve are exactly equal to those of (12) in the non-spiking network. However (12) will in general be applied to a corresponding input $\mathbf{y}$ where it may occur that for some

$j \in \{1, \ldots, m\}$ one has $y_i = 0$ for all $i \in G_j$ (since none of the neurons $y_i$ with $i \in G_j$ fired in the spiking network during $[t - 10\text{ms}, t]$). Hence we arrive at an application of (12) to an input $\mathbf{y}$ with missing attributes, as discussed in section 3.4.

Since several neurons $z_k$ are likely to fire during the presentation of an external input $\mathbf{x}$ (each hand-written digit was presented for 50ms in section 2; but a much shorter presentation time of 10ms also works quite well), this external input $\mathbf{x}$ gives in general rise to several applications of the STDP rule. This corresponds to several applications of rule (12) to the same input (but with different choices of missing attributes) in the non-spiking network. In the experiments in section 2, every example in the non-spiking network with missing attributes was therefore presented for 10 steps, such that the average number of learning steps is the same as in the spiking case. The learning process of the spiking network corresponds to a slight variation of the stochastic online EM algorithm that is implemented through (12) according to the analysis of section 3.3.

## 4 Discussion

The model for discovering hidden causes of inputs that is proposed in this extended abstract presents an interesting shortcut for implementing and learning generative models for input data in networks of neurons. Rather than building and adapting an explicit model for re-generating internally the distribution of input data, our approach creates an implicit model of the input distribution (see Fig. 3B) that is encoded in the weights of neurons in a simple WTA-circuit. One might call it a Vapnik-style [16] approach towards generative modeling, since it focuses directly on the task to represent the most likely hidden causes of the inputs through neuronal firing. As the theoretical analysis via non-spiking networks in section 3 has shown, this approach also offers a new perspective for generating self-adapting networks on the basis of traditional artificial neural networks. One just needs to add the stochastic and non-feedforward parts required for implementing stochastic WTA circuits to a 1-layer feedforward network, and apply the Hebbian learning rule (12) to the feedforward weights. One interesting aspect of the "implicit generative learning" approach that we consider in this extended abstract is that it retains important advantages of the generative learning approach, faster learning and better generalization [17], while retaining the algorithmic simplicity of the discriminative learning approach.

Our approach also provides a new method for analyzing details of STDP learning rules. The simulation results of section 2 show that a simplified STDP rule that can be understood clearly from the perspective of stochastic online EM with a suitable Hebbian learning rule, provides good performance in discovering hidden causes for a standard benchmark dataset. A more complex STDP rule, whose learning curve better matches experimentally recorded average changes of synaptic weights, provides almost the same performance. For a comparison of the STDP curves in Fig. 1B with experimentally observed STDP curves one should keep in mind, that most experimental data on STDP curves are for very low firing rates. The STDP curve of Fig. 7C in [18] for a firing rate of 20Hz has, similarly as the STDP curves in Fig. 1B of this extended abstract, no pronounced negative dip, and instead an almost constant negative part.

In our upcoming paper [8] we will provide full proofs for the results announced in this extended abstract, as well as further applications and extensions of the learning result. We will also demonstrate, that the learning rules that we have proposed are robust to noise, and that they are matched quite well by experimental data.

**Acknowledgments**

We would like to thank the anonymous reviewer for a hint in the notational formalism. Written under partial support by the Austrian Science Fund FWF, project # P17229-N04, project # S9102-N04, and project # FP6-015879 (FACETS) as well as # FP7-216593 (SECO) of the European Union.

## Footnotes

[1]A similar network of spiking neurons had been applied successfully in [7] to learn with STDP the classification of symbolic (i.e., not handwritten) characters. Possibly our theoretical analysis could also be used to explain their simulation result.

[2]Pixels were binarized to black/white. All pixels that were black in less than $5\%$ of the training examples were removed, leaving $m = 429$ external variables $x_j$, that were encoded by $n = 858$ spiking neurons $y_i$. Our approach works just as well for external variables $x_j$ that assume any finite number of values, provided that they are presented to the network through population coding with one variable $y_i$ for every possible value of $x_j$. In fact, the approach appears to work also for the commonly considered population coding of continuous external variables.

[3]This amounts to a representation of the EPSP caused by a firing of neuron $y_i$ by a step function, which facilitates the theoretical analysis in section 3. Learning with the spiking network works just as well for biologically realistic EPSP forms.

[4]Whereas the weights in the theoretical analysis of section 3 will approximate logs of probabilities (see (6)), one can easily make all weights non-negative by restricting the range of these *log*-probabilities to $[-5, 0]$, and then adding a constant 5 to all weight values. This transformation gives rise to the factor $c = e^5$ in Fig. 1B.

[5]Hence one can extend $p^*(\mathbf{y})$ for each fixed $\mathbf{w}$ to a joint distribution $p^*(\mathbf{y}, \mathbf{z})$, where the $\mathbf{z}$ are generated for each $\mathbf{y}$ by $\mathcal{N}_{\mathbf{w}}$.

[6]The equilibrium condition (13) only sets a necessary constraint for the the quotient of the two directions of the update in (12). The actual formulation of (12) is motivated by the goal of updating a sufficient statistics.

[7]In our experiments we used an adaptation of the variance tracking heuristic from [13]. If we assume that the consecutive values of the weights represent independent samples of their true stochastic distribution at the current learning rate, then this observed distribution is the log of a beta-distribution of the above mentioned parameters of the sufficient statistics. Analytically this distribution has the first and second moments $\mathrm{E}[w_{ki}] \approx \log \frac{a_{ki}}{N_i}$ and $\mathrm{E}[w_{ki}^2] \approx \mathrm{E}[w_{ki}]^2 + \frac{1}{a_{ki}} + \frac{1}{N_i}$, leading to the estimate $\eta_{ki}^{new} = \frac{1}{N_i} = \frac{\mathrm{E}[w_{ki}^2] - \mathrm{E}[w_{ki}]^2}{e^{-\mathrm{E}[w_{ki}]} + 1}$. The empirical estimates of these first two moments can be gathered online by exponentially decaying averages using the same learning rate $\eta_{ki}$.

## References

[1] Y. Dan and M. Poo. Spike timing-dependent plasticity of neural circuits. *Neuron*, 44:23–30, 2004.

[2] L. F. Abbott and S. B. Nelson. Synaptic plasticity: taming the beast. *Nature Neuroscience*, 3:1178–1183, 2000.

[3] A. Morrison, A. Aertsen, and M. Diesmann. Spike-timing-dependent plasticity in balanced random networks. *Neural Computation*, 19:1437–1467, 2007.

[4] R. J. Douglas and K. A. Martin. Neuronal circuits of the neocortex. *Annu Rev Neurosci*, 27:419–451, 2004.

[5] G. E. Hinton and Z. Ghahramani. Generative models for discovering sparse distributed representations. *Philos Trans R Soc Lond B Biol Sci.*, 352(1358):1177–1190, 1997.

[6] Y. LeCun, L. Bottou, Y. Bengio, and P. Haffner. Gradient-based learning applied to document recognition. *Proceedings of the IEEE*, 86(11):2278–2324, 1998.

[7] A. Gupta and L. N. Long. Character recognition using spiking neural networks. *IJCNN*, pages 53–58, 2007.

[8] B. Nessler, M. Pfeiffer, and W. Maass. Spike-timing dependent plasticity performs stochastic expectation maximization to reveal the hidden causes of complex spike inputs. *(in preparation)*.

[9] M. Meilă and D. Heckerman. An experimental comparison of model-based clustering methods. *Machine Learning*, 42(1):9–29, 2001.

[10] C. M. Bishop. *Pattern Recognition and Machine Learning*. Springer, New York, 2006.

[11] G. McLachlan and D. Peel. *Finite mixture models*. Wiley, 2000.

[12] J.H. Kushner and G.G. Yin. *Stochastic approximation algorithms and applications*. Springer, 1997.

[13] B. Nessler, M. Pfeiffer, and W. Maass. Hebbian learning of bayes optimal decisions. In *Advances in Neural Information Processing Systems 21*, pages 1169–1176. MIT Press, 2009.

[14] M. Sato. Fast learning of on-line EM algorithm. *Rapport Technique, ATR Human Information Processing Research Laboratories*, 1999.

[15] Z. Ghahramani and M.I. Jordan. Mixture models for learning from incomplete data. *Computational Learning Theory and Natural Learning Systems*, 4:67–85, 1997.

[16] V. Vapnik. Universal learning technology: Support vector machines. *NEC Journal of Advanced Technology*, 2:137–144, 2005.

[17] A. Y. Ng and M. I. Jordan. On discriminative vs. generative classifiers: A comparison of logistic regression and naive Bayes. *Advances in Neural Information Processing Systems (NIPS)*, 14:841–848, 2002.

[18] P. J. Sjöström, G. G. Turrigiano, and S. B. Nelson. Rate, timing, and cooperativity jointly determine cortical synaptic plasticity. *Neuron*, 32:1149–1164, 2001.

